# Coding efficiency and detectability of rate fluctuations with non-Poisson neuronal firing

**Shinsuke Koyama**[*]
Department of Statistical Modeling
The Institute of Statistical Mathematics
10-3 Midori-cho, Tachikawa, Tokyo 190-8562, Japan
skoyama@ism.ac.jp

## Abstract

Statistical features of neuronal spike trains are known to be non-Poisson. Here, we investigate the extent to which the non-Poissonian feature affects the efficiency of transmitting information on fluctuating firing rates. For this purpose, we introduce the Kullback-Leibler (KL) divergence as a measure of the efficiency of information encoding, and assume that spike trains are generated by time-rescaled renewal processes. We show that the KL divergence determines the lower bound of the degree of rate fluctuations below which the temporal variation of the firing rates is undetectable from sparse data. We also show that the KL divergence, as well as the lower bound, depends not only on the variability of spikes in terms of the coefficient of variation, but also significantly on the higher-order moments of interspike interval (ISI) distributions. We examine three specific models that are commonly used for describing the stochastic nature of spikes (the gamma, inverse Gaussian (IG) and lognormal ISI distributions), and find that the time-rescaled renewal process with the IG distribution achieves the largest KL divergence, followed by the lognormal and gamma distributions.

## 1 Introduction

Characterizing the statistical features of spike time sequences in the brain is important for understanding how the brain represents information about stimuli or actions in the sequences of spikes. Although the spike trains recorded from *in vivo* cortical neurons are known to be highly irregular [20, 24], a recent non-stationary analysis has revealed that individual neurons signal with non-Poisson firing, the characteristics of which are strongly correlated with the function of the cortical area [21].

This raises the question of what the neural coding advantages of non-Poisson spiking are. It could be that the precise timing of spikes carries additional information about the stimuli or actions [6, 15]. It is also possible that the efficiency of transmitting fluctuating rates might be enhanced by non-Poisson firing [5, 17]. Here, we explore the latter possibility.

In the problem of estimating firing rates, there is a minimum degree of rate fluctuation below which a rate estimator cannot detect the temporal variation of the firing rate [23]. If, for instance, the degree of temporal variation of the rate is on the same order as that of the noise, a constant rate might be chosen as the most likely estimate for a given spike train. It is, therefore, interesting to see how the minimum degree of rate fluctuation depends on the non-Poissonian feature of spike trains.

In this study, we investigate the extent to which the non-Poissonian feature of spike trains affects the encoding efficiency of rate fluctuations. In addition, we address the question of how the de-

---

[*]http://skoyama.blogspot.jp

tectability of rate fluctuations depends on the encoding efficiency. For this purpose, we introduce the Kullback-Leibler (KL) divergence to measure the encoding efficiency, and assume that spike sequences are generated by time-rescaled renewal processes. With the aid of analytical and numerical studies, we suggest that the lower bound of detectable rate fluctuations, below which the empirical Bayes decoder cannot detect the rate fluctuations, is uniquely determined by the KL divergence. By examining three specific models (the time-rescaled renewal process with the gamma, inverse Gaussian (IG) and lognormal interspike interval (ISI) distributions), it is shown that the KL divergence, as well as the lower bound, depends not only on the first- and second-order moments, but also significantly on the higher-order moments of the ISI distributions. We also find that among the three ISI distributions, the IG distribution achieves the highest efficiency of coding information on rate fluctuations.

## 2 Encoding rate fluctuations using time-rescaled renewal processes

**Definitions of time-rescaled renewal processes and KL divergence**

We introduce time-rescaled renewal processes for a model of neuronal spike trains constructed in the following way. Let $f_\kappa(y)$ be a family of ISI distributions with the unit mean (i.e., $\int_0^\infty y f_\kappa(y) dy = 1$), where $\kappa$ controls the shape of the distribution, and $\lambda(t)$ be a fluctuating firing rate. A sequence of spikes $\{t_i\} := \{t_1, t_2, \ldots, t_n\}$ is generated in the following steps: (i) Derive ISIs $\{y_1, y_2, \ldots, y_n\}$ independently from $f_\kappa(y)$, and arrange the ISIs sequentially to form a spike train of the unit rate; $i$th spike is given by summing the previous ISIs as $s_i = \sum_{j=1}^{i} y_j$. (ii) Transform $\{s_1, s_2, \ldots, s_n\}$ to $\{t_1, t_2, \ldots, t_n\}$ according to $t_i = \Lambda^{-1}(s_i)$, where $\Lambda^{-1}(s_i)$ is the inverse of the function $\Lambda(t) = \int_0^t \lambda(u) du$. This transformation ensures that the instantaneous firing rate of $\{t_i\}$ corresponds to $\lambda(t)$, while the shape of the ISI distribution $f_\kappa(y)$, which characterizes the firing irregularity, is unchanged in time. This is in agreement with the empirical fact that the degree of irregularity in neuronal firing is generally maintained in cortical processing [21, 22], while the firing rate $\lambda(t)$ changes in time. The probability density of the occurrence of spikes at $\{t_i\}$ is, then, given by

$$p_\kappa(\{t_i\}|\{\lambda(t)\}) = \prod_{i=1}^{n} \lambda(t_i) f_\kappa(\Lambda(t_i) - \Lambda(t_{i-1})). \tag{1}$$

where $t_0 = 0$.

We next introduce the KL divergence for measuring the encoding efficiency of fluctuating rates. For this purpose, we assume that $\lambda(t)$ is ergodic with a stationary distribution $p(\lambda)$, the mean of which is given by $\mu$:

$$\langle \lambda \rangle_\lambda := \int_0^\infty \lambda p(\lambda) d\lambda = \lim_{T \to \infty} \frac{1}{T} \int_0^T \lambda(t) dt = \mu. \tag{2}$$

Consider a probability density of a renewal process that has the same ISI density $f_\kappa(x)$ and the constant rate $\mu$:

$$p_\kappa(\{t_i\}|\mu) = \prod_{n=1}^{n} \mu f_\kappa(\mu(t_i - t_{i-1})). \tag{3}$$

The KL divergence between $p_\kappa(\{t_i\}|\{\lambda(t)\})$ and $p_\kappa(\{t_i\}|\mu)$ is, then, defined as

$$D_\kappa(\lambda(t)||\mu) \quad := \quad \lim_{T \to \infty} \sum_{n=0}^{\infty} \frac{1}{T} \int_0^T \int_{t_1}^T \cdots \int_{t_{n-1}}^T p_\kappa(\{t_i\}|\{\lambda(t)\})$$
$$\times \log \frac{p_\kappa(\{t_i\}|\{\lambda(t)\})}{p_\kappa(\{t_i\}|\mu)} dt_1 dt_2 \cdots dt_n. \tag{4}$$

Since it is defined as the entropy of a renewal process with the fluctuating rate $\lambda(t)$ relative to that with the constant rate $\mu$, $D_\kappa(\lambda(t)||\mu)$ can be interpreted as the amount of information on the rate fluctuations encoded into spike trains. Note that a similar quantity has been introduced in [3], where the quantity was computed only under a Poisson model.

Substituting Eqs. (1) and (3) into Eq. (4) and further assuming ergodicity of spike trains, the KL divergence can be expressed as

$$
\begin{aligned}
D_\kappa(\lambda(t)||\mu) &= \lim_{n\to\infty} \frac{1}{t_n - t_0} \log \frac{p_\kappa(\{t_i\}|\{\lambda(t)\})}{p_\kappa(\{t_i\}|\mu)} \\
&= \lim_{n\to\infty} \frac{1}{t_n - t_0} \sum_{i=1}^{n} \big\{ \log \lambda(t_i) + \log f_\kappa(\Lambda(t_i) - \Lambda(t_{i-1})) \\
&\quad - \log \mu - \log f_\kappa(\mu(t_i - t_{i-1})) \big\}.
\end{aligned}
\tag{5}
$$

This expression can be used for computing the KL divergence numerically by simulating a large number of spikes $n \gg 1$.

**Three ISI distributions and their KL divergence**

In order to examine the behavior of the KL divergence, we use the three specific ISI distributions for $f_\kappa(y)$ (the gamma, inverse Gaussian (IG) and lognormal distributions), which have been used to describe the stochastic nature of ISIs [9, 10, 14]. These distributions and their coefficient of variation ($C_V = \sqrt{Var(X)}/E(X)$) are given by

$$
\text{gamma} \quad : \quad f_\kappa(y) = \kappa^\kappa y^{\kappa-1} e^{-\kappa y} / \Gamma(\kappa), \quad C_V = 1/\sqrt{\kappa},
\tag{6}
$$

$$
\text{IG} \quad : \quad f_\kappa(y) = \sqrt{\frac{\kappa}{2\pi y^3}} \exp\left[ -\frac{\kappa(y-1)^2}{2y} \right], \quad C_V = 1/\sqrt{\kappa},
\tag{7}
$$

$$
\text{lognormal} \quad : \quad f_\kappa(y) = \frac{1}{y\sqrt{2\pi\kappa}} \exp\left[ -\frac{(\log y + \frac{\kappa}{2})^2}{2\kappa} \right], \quad C_V = \sqrt{e^\kappa - 1},
\tag{8}
$$

where $\Gamma(\kappa) = \int_0^\infty x^{\kappa-1} e^{-x} dx$ is the gamma function. Figure 1a illustrates the shape of the three distributions with three different values of $C_V$.

The KL divergence for the three models is analytically solvable when the rate fluctuation has a long time scale relative to the mean ISI. Here, we show the derivation for the gamma distribution. (The derivations for the IG and lognormal distributions are essentially the same.) Inserting Eq. (6) into Eq. (5) leads to

$$
\begin{aligned}
D_\kappa(\lambda(t)||\mu) &= \lim_{n\to\infty} \frac{1}{t_n - t_0} \sum_{i=1}^{n} \big\{ \log \lambda(t_i) + (\kappa - 1) \log[\Lambda(t_i) - \Lambda(t_{i-1})] \\
&\quad - (\kappa - 1) \log(t_i - t_{i-1}) \big\} - \kappa\mu \log\mu,
\end{aligned}
\tag{9}
$$

where we used $\frac{1}{t_n - t_0} \int_{t_0}^{t_n} \lambda(t) dt \to \mu$ and $\frac{n}{t_n - t_0} \to \mu$ as $n \to \infty$. By introducing the "averaged" firing rate in the $i$th ISI: $\bar{\lambda}_i := \frac{\Lambda(t_i) - \Lambda(t_{i-1})}{t_i - t_{i-1}}$, we obtain $\log[\Lambda(t_i) - \Lambda(t_{i-1})] = \log \bar{\lambda}_i + \log(t_i - t_{i-1})$. Assuming that the time scale of the rate fluctuation is longer than the mean ISI so that $\bar{\lambda}_i$ is approximated to $\lambda(t_i)$, Eq. (9) becomes

$$
\begin{aligned}
D_\kappa(\lambda(t)||\mu) &= \kappa \lim_{n\to\infty} \frac{1}{t_n - t_0} \sum_{i=1}^{n} \log \lambda(t_i) - \kappa\mu \log\mu \\
&= \kappa \left\{ \lim_{T\to\infty} \frac{1}{T} \int_0^T \sum_i \delta(t - t_i) \log \lambda(t) dt - \mu \log\mu \right\}.
\end{aligned}
\tag{10}
$$

The fluctuation in the apparent spike count is given by the variance to mean ratio as represented by the Fano factor [8]. For the renewal process in which ISIs are drawn from a given distribution function, it is proven that the Fano factor is related to the ISI variability with $F \approx C_V^2$ [4]. Thus, for a long range time scale in which a serial correlation of spikes is negligible, the spike train in Eq. (10) can be approximated to

$$
\sum_{i=1}^{n} \delta(t - t_i) \approx \lambda(t) + \sqrt{\lambda(t)/\kappa} \xi(t),
\tag{11}
$$

where $\xi(t)$ is a fluctuating process such that $\langle \xi(t) \rangle = 0$ and $\langle \xi(t)\xi(t') \rangle = \delta(t - t')$. Using this, the first term on the rhs of (10) can be evaluated as

$$\lim_{T \to \infty} \frac{1}{T} \int_0^T \lambda(t) \log \lambda(t) dt + \lim_{T \to \infty} \frac{1}{T} \int_0^T \sqrt{\lambda(t)/\kappa} \log \lambda(t) \xi(t) dt \quad = \quad \langle \lambda \log \lambda \rangle_\lambda, \quad (12)$$

where the second term on the lhs has vanished due to a property of stochastic integrals. Therefore, the KL divergence of the gamma distribution is obtained as

$$D_\kappa(\lambda(t)\|\mu) \quad = \quad \kappa\big\{\langle \lambda \log \lambda \rangle_\lambda - \mu \log \mu \big\}. \quad (13)$$

In the same way, the KL divergence for the IG and lognormal distributions are, respectively, derived as

$$D_\kappa(\lambda(t)\|\mu) \quad = \quad \frac{\mu}{2} \log \mu - \frac{1}{2}\langle \lambda \log \lambda \rangle_\lambda + \frac{\kappa+1}{2\mu}\langle (\lambda - \mu)^2 \rangle_\lambda, \quad (14)$$

and

$$D_\kappa(\lambda(t)\|\mu) = \frac{\mu}{2\kappa}(\log \mu)^2 - \frac{\log \mu}{\kappa}\langle \lambda \log \lambda \rangle_\lambda + \frac{1}{2\kappa}\langle \lambda(\log \lambda)^2 \rangle_\lambda. \quad (15)$$

See the supplementary material for the details of their derivations.

### Results

We compute the KL divergence for the three models, in which the rate fluctuates according to the Ornstein-Uhlenbeck process. Formally, the rate process is given by $\lambda(t) = [x(t)]_+$, where $[\cdot]_+$ is the rectification function:

$$[x]_+ = \begin{cases} x, & x > 0 \\ 0, & \text{otherwise} \end{cases} \quad (16)$$

and $x(t)$ is derived from the Ornstein-Uhlenbeck process:

$$\frac{dx(t)}{dt} = -\frac{x(t) - \mu}{\tau} + \sigma\sqrt{\frac{2}{\tau}}\xi(t), \quad (17)$$

where $\xi(t)$ is the Gaussian white noise.

Figure 1b depicts the KL divergence as a function of $\sigma$ for $C_V$=0.6, 1 and 1.5. The analytical results (the solid lines) are in good agreement with the numerical results (the error bars). The KL divergence for the three models increases as $\sigma$ is increased and as $C_V$ is decreased, which is rather obvious since larger $\sigma$ and smaller $C_V$ imply lower noise entropy of spike trains. One nontrivial result is that, even if the three models share the same values of $\sigma$ and $C_V$, the KL divergence of each model significantly differs from that of the others: the IG distribution achieves the largest KL divergence, followed by the lognormal and gamma distributions. The difference in the KL divergence among the three models becomes larger as $C_V$ grows larger. Since the three models share the same firing rate $\lambda(t)$ and $C_V$, it can be concluded that the higher-order (more than second-order) moments of ISI distributions strongly affect the KL divergence.

In order to confirm this result for another rate process, we examine a sinusoidal rate process, $\lambda(t) = \mu + \sigma \sin t/\tau$, and observe the same behavior as the Ornstein-Uhlenbeck rate process (Figure 1c).

## 3 Decoding fluctuating rates using the empirical Bayes method

In this section, we show that the KL divergence (4) determines the lower bound of the degree of rate fluctuation below which the empirical Bayes estimator cannot detect rate fluctuations.

### The empirical Bayes method

We consider decoding a fluctuation rate $\lambda(t)$ from a given spike train $\{t_i\} := \{t_1 \ldots, t_n\}$ in an observation interval $[0, T]$ by the empirical Bayes method. Let $x(t) \in \mathbb{R}$ be a latent variable that

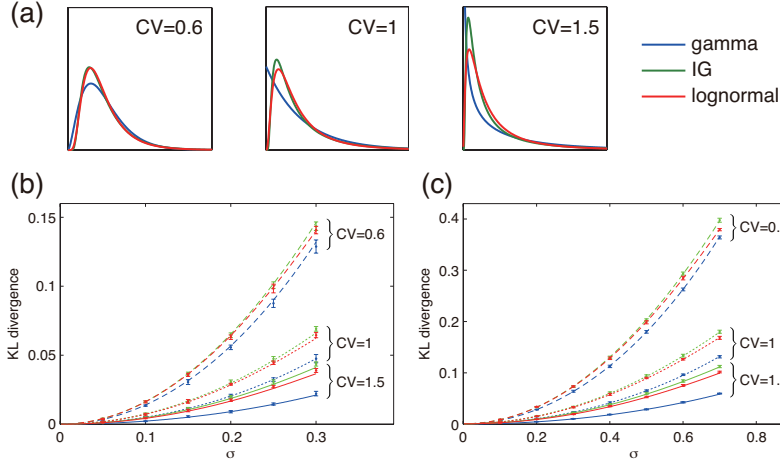

Figure 1: (a) The gamma (blue), IG (green) and lognormal (red) ISI distribution functions for $C_V$=0.6, 1 and 1.5. (b) The KL divergence as a function of $\sigma$ for $C_V$=0.6, 1 and 1.5, when the rate fluctuates according to the Ornstein-Uhlenbeck process (17) with $\mu = 1$ and $\tau = 10$. The blue, green and red indicate the KL divergence for the gamma, IG and lognormal distribution, respectively. The lines represent the theoretical values obtained by Eqs. (13), (14) and (15), and the error bars represent the average and standard deviation numerically computed according to Eq. (5) with $n = 50,000$ and 10 trials. (c) The KL divergence for the sinusoidally modulated rate, $\lambda(t) = \mu + \sigma \sin t / \tau$, with $\mu = 1$ and $\tau = 10$.

is transformed from $\lambda(t)$ via the log-link function $x(t) = \log \lambda(t)$. For the inference of $\lambda(t)$ from $\{t_i\}$, we use a prior distribution of $x(t)$, such that the large gradient of $x(t)$ is controlled by

$$p_\gamma(\{x(t)\}) \propto \exp \left[ -\frac{1}{2\gamma^2} \int_0^T \left( \frac{dx(t)}{dt} \right)^2 dt \right], \tag{18}$$

where the hyperparameter $\gamma$ controls the roughness of the latent process $x(t)$: with the small $\gamma$, the model requires a constant latent process, and vice versa. By inverting the conditional probability distribution with the Bayes' theorem, the posterior distribution of $\{x(t)\}$ is obtained as

$$p_{\kappa,\gamma}(\{x(t)\}|\{t_i\}) = \frac{p_\kappa(\{t_i\}|\{x(t)\})p_\gamma(\{x(t)\})}{p_{\kappa,\gamma}(\{t_i\})}. \tag{19}$$

The hyperparameters, $\gamma$ and $\kappa$, which represent the roughness of the latent process and the shape of the ISI density function, can be determined by maximizing the marginal likelihood [16] defined by

$$p_{\kappa,\gamma}(\{t_i\}) = \int p_\kappa(\{t_i\}|\{x(t)\})p_\gamma(\{x(t)\})\mathcal{D}\{x(t)\}, \tag{20}$$

where $\int \mathcal{D}\{x(t)\}$ represents the integration over all possible latent process paths. Under a set of hyperparameters $\hat{\gamma}$ and $\hat{\kappa}$ that are determined by the marginal likelihood maximization, we can determine the maximum a posteriori (MAP) estimate of the latent process $\hat{x}(t)$. The method for implementing the empirical Bayes analysis is summarized in the Appendix.

**Detectability of rate fluctuations**

We first examine the gamma distribution (6). For synthetic spike trains ($n = 1,000$) generated by the time-rescaled renewal process with the gamma ISI distribution, in which the rate fluctuates according to the Ornstein-Uhlenbeck process (17) with $\mu = 1$ and $\tau = 10$, we attempt to decode $\lambda(t)$ using the empirical Bayes decoder. Depending on the amplitude of the rate fluctuation $\sigma$ and $C_V$ of $f_\kappa(y)$, the empirical Bayes decoder provides qualitatively two distinct rate estimations: (I) a fluctuating rate estimation ($\hat{\gamma} > 0$) for large $\sigma$ and small $C_V$, or (II) a constant rate estimation ($\hat{\gamma} = 0$) for small $\sigma$ and large $C_V$ (Figure 2a). When $\sigma$ is increased or $C_V$ is decreased, the

empirical Bayes estimator exhibits a phase transition corresponding to the switch of the most likely rate estimation from (II) to (I) (Figure 2b). Note that below the critical point of this phase transition, the empirical Bayes method provides a constant rate as the most likely estimation even if the true rate process fluctuates. The critical point, thus, gives the lower bound for the degree of detectable rate fluctuations. It is also confirmed, using numerical simulations, that the phase transition occurs not only with the gamma distribution, but also with the IG and lognormal distributions (Figure 2c,d).

For the time-rescaled renewal process with the gamma ISI distribution, we could analytically derive the formula that the lower bound satisfies as:

$$D_\kappa(\lambda(t)||\mu) = \frac{\phi(0)}{4 \max_\eta \int_0^\infty \phi(u)e^{-\eta u}du}, \tag{21}$$

where $\phi(u)$ is the correlation function of $\lambda(t)$. (See supplementary material for the derivation.) Eq. (21) is in good agreement with the simulation result for the entire parameter space (the solid line in Figure 2a).

The expression of Eq. (21) itself does not depend on the gamma distribution. We investigated if this formula is also applicable to the IG and lognormal distributions, and found that the theoretical lower bounds (the solid lines in Figure 2c,d) indeed do correspond to those obtained by the numerical simulations; this result implies that Eq. (21) is applicable to more general time-rescaled renewal processes.

Figure 2e compares the lower bounds among the three distributions. The lower bound of the IG distribution is the lowest, followed by the lognormal and gamma distributions, which is expected from the result in Figure 1b, as the lower bound is identically determined by the KL divergence via Eq. (21).

We also examined the sinusoidally modulated rate, $\lambda(t) = \mu + \sigma \sin t/\tau$; the qualitative result remains the same (Figure 2f-h).

## 4  Discussion

In this study, we first examined the extent to which spike trains derived from time-rescaled renewal processes encode information on fluctuating rates. The encoding efficiency is measured by the KL divergence between two renewal processes with fluctuating and constant rates. We showed that the KL divergence significantly differs among the gamma, IG and lognormal ISI distributions, even if these three processes share the same rate fluctuation $\lambda(t)$ and $C_V$ (Figure 1b). This suggests that the higher-order moments of ISIs play an important role in encoding information on fluctuating rates. Among the three distributions, the IG distribution achieves the largest KL divergence, followed by the lognormal and gamma distributions. A similar result has been reported for stationary renewal processes [12].

Since the KL divergence gives the distance between two probability distributions, Eq. (4) is naturally related to the ability to discriminate between a fluctuating rate and a constant rate. In fact, the lower bound of the degree of rate fluctuation, below which the empirical Bayes decoder cannot discriminate the underlying fluctuating rate from a constant rate, satisfies the formula (21). There commonly exists a lower bound below which the underlying rate fluctuations are undetectable, not only in the empirical Bayes method with the above prior distribution (18), but also with other prior distributions, and in other rate estimators such as a time-histogram. The lower bound in these methods has been derived for inhomogeneous Poisson processes as $\tau\sigma^2/\mu \sim O(1)$, where $\tau$, $\sigma$ and $\mu$ are the time scale, amplitude and mean of the rate fluctuation, respectively [23]. Thus, Eq. (21), or equivalently $\tau D_\kappa(\lambda(t)||\mu) \sim O(1)$ is regarded as a generalization to the non-Poisson processes. Here, the crucial step for this generalization is incorporating the KL divergence into the formula.

Note that the formula (21) was derived analytically under the assumption of the gamma ISI distribution, and then was shown to hold for the IG and lognormal ISI distributions with numerical simulations. The analytical tractability of the gamma family lies in the fact that it is the only scale family that admits the mean as a sufficient statistic. We conjecture, from our results with the three specific models, that Eq. (21) is applicable to more general time-rescaled renewal processes (even to "non-renewal" processes), which is open to future research.

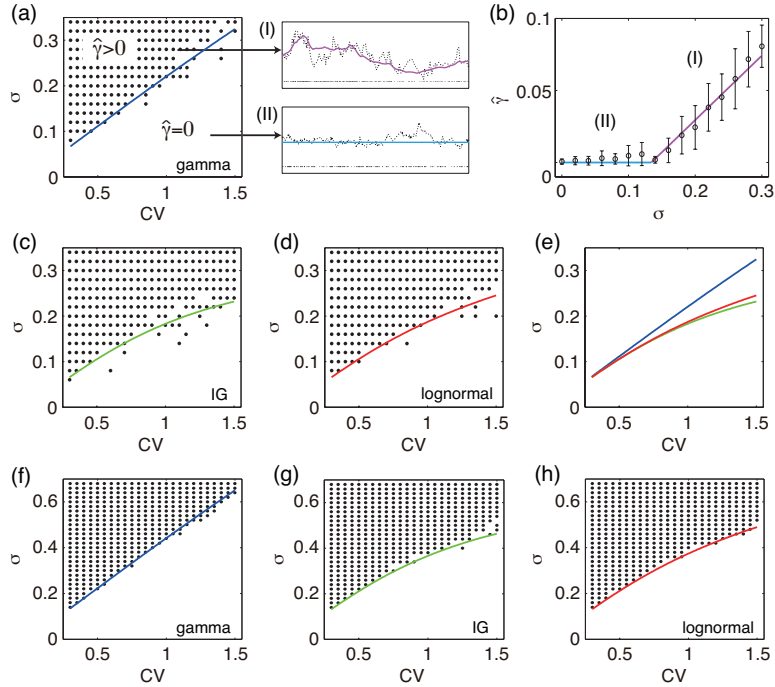

Figure 2: (a) Left: the phase diagram for sequences generated by the time-rescaled renewal process with the gamma ISI distribution. The ordinate represents the amplitude of rate fluctuation $\sigma$, and abscissa represents $C_V$ of the gamma ISI distribution. The dots represent the result of numerical simulations in which the empirical Bayes decoder provides a fluctuating rate estimation ($\hat{\gamma} > 0$). Each dot is plotted if $\hat{\gamma} > 0$ in more than 20 out of 40 identical trials. The solid line represents the theoretical lower bound obtained by the formula (21). Right: raster plots of sample spike trains and the estimated rates. The dotted lines and the solid lines represent the underlying rates and the estimated rates, respectively. The parameters $(C_V, \sigma)$ of top ($\hat{\gamma} > 0$) and bottom ($\hat{\gamma} = 0$) are $(0.6, 0.3)$ and $(1.5, 0.15)$, respectively. (b) The optimal hyperparameter $\hat{\gamma}$ as a function of $\sigma$ for $C_V = 0.6$. The solid line represents the theoretical value, and the error bars represent the average and standard deviation of $\hat{\gamma}$ determined by applying the empirical Bayes algorithm to 40 trials. (c, d) The phase diagrams for the IG and lognormal ISI distributions. (e) Comparison of the lower bounds among the three models. (f-h) The phase diagrams for the gamma, IG and lognormal ISI distributions, when the rate process is given by $\lambda(t) = \mu + \sigma \sin t/\tau$ with $\mu = 1$ and $\tau = 10$.

A recent non-stationary analysis has revealed that individual neurons in the cortex signal with non-Poisson firing, which has empirically been characterized by measures based on the second-order moment of ISIs, such as $C_V$ and $L_V$ [21, 22]. Our results, however, suggest that it may be important to take into account the higher-order moments of ISIs for characterizing "irregularity" of cortical firing, in order to gain information on fluctuating firing rates. It has also been demonstrated that using non-Poisson spiking models enhances the performance of neural decoding [2, 11, 19]. Our results provide theoretical support for this as well.

## Appendix: Implementation of the empirical Bayes method

### Discretization

To construct a practical algorithm for performing empirical Bayes decoding, we first divide the time axis into a set of intervals $(t_{i-1}, t_i]$ $(i = 1, \ldots, n)$. We assume that the firing rate within each interval $(t_{i-1}, t_i]$ does not change drastically (which is a reasonable assumption in practice), so that it can be approximated to a constant value $\lambda_i$. Letting $T_i = t_i - t_{i-1}$ be the $i$th ISI, the probability density of $\{T_i\} \equiv \{T_1, T_2, \ldots, T_n\}$, given the rate process $\{\lambda_i\} \equiv \{\lambda_1, \lambda_2 \ldots, \lambda_n\}$

is obtained from Eq. (1) as $p_\kappa(\{T_i\}|\{\lambda_i\}) = \prod_{i=1}^n \lambda_i f_\kappa(\lambda_i T_i)$. The rate process is linked with the latent process via $x_i = \log \lambda_i$. With the same time-discretization, the prior distribution of the latent process $\{x_i\} \equiv \{x_1, x_2, \ldots, x_n\}$, which corresponds to Eq. (18), is derived as $p_\gamma(\{x_i\}) = p(x_1) \prod_{i=2}^n p_\gamma(x_i|x_{i-1})$, where

$$p_\gamma(x_i|x_{i-1}) = \frac{1}{\sqrt{\pi\gamma^2(T_i + T_{i-1})}} \exp\left[ -\frac{(x_i - x_{i-1})^2}{\gamma^2(T_i + T_{i-1})} \right], \tag{22}$$

and $p(x_1)$ is the probability density function of the initial latent rate variable.

$p(\{T_i\}|\{\lambda_i\})$ and $p_\gamma(\{x_i\})$ define a discrete-time state space model. We note that this provides a good approximation to the original continuous-time model if the timescale of the rate fluctuation is larger than the mean ISI.

## EM algorithm

We assume that the ISI density function can be rewritten in the form of exponential family distributions with respect to the shape parameter $\kappa$:

$$p_\kappa(T_i|\phi_i) := \lambda_i f_\kappa(\lambda_i T_i) = \exp[\kappa S(T_i, \phi_i) - \varphi(\kappa) + c(T_i, \phi_i)], \tag{23}$$

with an appropriate parameter representation $\phi_i = \phi(\lambda_i, \kappa)$. Here, $\kappa$ is the natural parameter of the exponential family and $S(T_i, \phi_i)$ is its sufficient statistic. Suppose that the potential $\varphi(\kappa)$ is a convex function. The expectation of $S(T_i, \phi_i)$ is then given by

$$\eta = \int S(T_i, \phi_i) p_\kappa(T_i|\phi_i) dT_i = \frac{d\varphi(\kappa)}{d\kappa}. \tag{24}$$

Since $\varphi(\kappa)$ is convex, there is one-to-one correspondence between $\kappa$ and $\eta$, and thus $\eta$ provides alternative parametrization to $\kappa$ [1]. The gamma (6), IG (7) and lognormal (8) distributions are included in this family.

With the parameterization $\eta$, the EM algorithm for the state space model is derived as follows. Suppose that we have estimations $\hat{\eta}_{(m)}$ and $\hat{\gamma}_{(m)}$ at the $m$th iteration. The estimations at the $(m+1)$th iteration are given by

$$\hat{\eta}_{(m+1)} = \frac{1}{n} \sum_{i=1}^n \langle S(T_i, \phi(x_i)) \rangle_{(m)}, \tag{25}$$

and

$$\hat{\gamma}^2_{(m+1)} = \frac{2}{n-1} \sum_{i=2}^n \frac{\langle (x_i - x_{i-1})^2 \rangle_{(m)}}{T_i + T_{i-1}}, \tag{26}$$

where $\langle\ \rangle_{(m)}$ denotes the expectation with respect to the posterior probability of $\{x_i\}$, given $\{T_i\}$, $\hat{\eta}_{(m)}$ and $\hat{\gamma}_{(m)}$. The posterior probability is computed by the Laplace approximation, introduced below. We update $\hat{\eta}$ and $\hat{\gamma}$ until the estimations converge. The estimation of $\kappa$ is then transformed from $\hat{\eta}$ with Eq. (24).

## Laplace approximation

We employ Laplace's method to compute an approximate posterior distribution of $\{x_i\}$. Let $\boldsymbol{x} = (x_1, x_2, \ldots, x_n)^t$ be the column vector of the latent process, $(\ )^t$ being the transpose of a vector. The MAP estimate of the latent process is obtained by maximizing the log posterior distribution:

$$l(\boldsymbol{x}) = \log p(x_1) + \sum_{i=2}^n \log p_\gamma(x_i|x_{i-1}) + \sum_{i=1}^n \log p_\kappa(T_i|x_i) + \text{const.}, \tag{27}$$

with respect to $\boldsymbol{x}$. We use a diffuse prior for $p(x_1)$ so that its contribution vanishes [7]. If $p_\gamma(x_i|x_{i-1})$ is log-concave in $x_i$ and $x_{i-1}$, and the $p_\kappa(T_i|x_i)$ is also log-concave in $x_i$, computing the MAP estimate is a concave optimization problem [18], which can be solved efficiently by a Newton method. Due to the Markovian Structure of the state-space model, the Hessian matrix, $J(\boldsymbol{x}) \equiv \nabla\nabla_{\boldsymbol{x}} l(\boldsymbol{x})$, becomes a tridiagonal matrix, which allows us to compute the Newton step in $O(n)$ time [13]. Let $\hat{\boldsymbol{x}}$ denote the MAP estimation of the posterior probability. The posterior probability is then approximated to a Gaussian whose mean vector and covariance matrix are given by $\hat{\boldsymbol{x}}$ and $-J(\hat{\boldsymbol{x}})^{-1}$, respectively.

**Acknowledgments**

This work was supported by JSPS KAKENHI Grant Number 24700287.

**References**

[1] S. Amari and H. Nagaoka. *Methods of Information Geometry*. Oxford University Press, 2000.

[2] R. Barbieri, M. C. Quirk, L. M. Frank, M. A. Wilson, and E. N. Brown. Construction and analysis of non-poisson stimulus-response models of neural spiking activity. *Journal of Neuroscience Methods*, 105:25–37, 2001.

[3] N. Brenner, S. P. Strong, R. Koberle, and W. Bialek. Synergy in a neural code. *Neural Computation*, 12:1531–1552, 2000.

[4] D. R. Cox. *Renewal Theory*. Chapman and Hal, 1962.

[5] J. P. Cunningham, B. M. Yu, K. V. Shenoy, and M. Sahani. Inferring neural firing rates from spike trains using Gaussian processes. In *Neural Information Processing Systems*, volume 20, pages 329–336, 2008.

[6] R. M. Davies, G. L. Gerstein, and S. N. Baker. Measurement of time-dependent changes in the irregularity of neural spiking. *Journal of Neurophysiology*, 96:906–918, 2006.

[7] J. Durbin and S. J. Koopman. *Time Series Analysis by State Space Methods*. Oxford University Press, 2001.

[8] U. Fano. Ionization yield of radiations. ii. the fluctuations of the number of ions. *Physical Review*, 72:26–29, 1947.

[9] G. L. Gerstein and B. Mandelbrot. Random walk models for the spike activity of a single neuron. *Biophysical Journal*, 4:41–68, 1964.

[10] S. Ikeda and J. H. Manton. Capacity of a single spiking neuron channel. *Neural Computation*, 21:1714–1748, 2009.

[11] A. L. Jacobs, G. Fridman, R. M. Douglas, N. M. Alam, P. E. Latham, G. T. Prusky, and S. Nirenberg. Ruling out and ruling in neural codes. *Proceedings of the National Academy of Sciences*, 106:5936–5941, 2009.

[12] K. Kang and S. Amari. Discrimination with spike times and ISI distributions. *Neural Computation*, 20:1411–1426, 2008.

[13] S. Koyama and L. Paninski. Efficient computation of the maximum a posteriori path and parameter estimation in integrate-and-fire and more general state-space models. *Journal of Computational Neuroscience*, 29:89–105, 2009.

[14] M. W. Levine. The distribution of the intervals between neural impulses in the maintained discharges of retinal ganglion cells. *Biological Cybernetics*, 65:459–467, 1991.

[15] B. N. Lundstrom and A. L. Fairhall. Decoding stimulus variance from a distributional neural code of interspike intervals. *Journal of Neuroscience*, 26:9030–9037, 2006.

[16] D. J. C. MacKay. Bayesian interpolation. *Neural Computation*, 4:415–447, 1992.

[17] T. Omi and S. Shinomoto. Optimizing time histograms for non-Poisson spike trains. *Neural Computation*, 23:3125–3144, 2011.

[18] L. Paninski. Log-concavity results on gaussian process methods for supervised and unsupervised learning. In *Neural Information Processing Systems*, volume 17, pages 1025–1032, 2005.

[19] J. W. Pillow, L. Paninski, V. J. Uzzell, E. P. Simoncelli, and E. J. Chichilnisky. Prediction and decoding of retinal ganglion cell responses with a probabilistic spiking model. *Journal of Neuroscience*, 23:11003–11013, 2005.

[20] M. N. Shadlen and W. T. Newsome. The variable discharge of cortical neurons: Implications for connectivity, computation, and information coding. *Journal of Neuroscience*, 18:3870–3896, 1998.

[21] S. Shinomoto, H. Kim, T. Shimokawa, N. Matsuno, S. Funahashi, K. Shima, I. Fujita, H. Tamura, T. Doi, K. Kawano, N. Inaba, K. Fukushima, S. Kurkin, K. Kurata, M. Taira, K. Tsutsui, H. Komatsu, T. Ogawa, K. Koida, J. Tanji, and K. Toyama. Relating neuronal firing patterns to functional differentiation of cerebral cortex. *PLoS Computational Biology*, 5:e1000433, 2009.

[22] S. Shinomoto, K. Shima, and J. Tanji. Differences in spiking patterns among cortical neurons. *Neural Computation*, 15:2823–2842, 2003.

[23] T. Shintani and S. Shinomoto. Detection limit for rate fluctuations in inhomogeneous poisson processes. *Physical Review E*, 85:041139, 2012.

[24] W. R. Softky and C. Koch. The highly irregular firing of cortical cells is inconsistent with temporal integration of random EPSPs. *Journal of Neuroscience*, 13:334–350, 1993.

